# Mechanism of neural interference by transcranial magnetic stimulation: network or single neuron?

**Yoichi Miyawaki**
RIKEN Brain Science Institute
Wako, Saitama 351-0198, JAPAN
yoichi_miyawaki@brain.riken.jp

**Masato Okada**
RIKEN Brain Science Institute
PRESTO, JST
Wako, Saitama 351-0198, JAPAN
okada@brain.riken.jp

## Abstract

This paper proposes neural mechanisms of transcranial magnetic stimulation (TMS). TMS can stimulate the brain non-invasively through a brief magnetic pulse delivered by a coil placed on the scalp, interfering with specific cortical functions with a high temporal resolution. Due to these advantages, TMS has been a popular experimental tool in various neuroscience fields. However, the neural mechanisms underlying TMS-induced interference are still unknown; a theoretical basis for TMS has not been developed. This paper provides computational evidence that inhibitory interactions in a neural population, not an isolated single neuron, play a critical role in yielding the neural interference induced by TMS.

## 1 Introduction

Transcranial magnetic stimulation (TMS) is an experimental tool for stimulating neurons via brief magnetic pulses delivered by a coil placed on the scalp. TMS can non-invasively interfere with neural functions related to a target cortical area with high temporal accuracy. Because of these unique and powerful features, TMS has been popular in various fields, including cognitive neuroscience and clinical application. However, despite its utility, the mechanisms of how TMS stimulates neurons and interferes with neural functions are still unknown. Although several studies have modeled spike initiation and inhibition with a brief magnetic pulse imposed on an isolated single neuron [1][2], it is rather more plausible to assume that a large number of neurons are stimulated massively and simultaneously because the spatial extent of the induced magnetic field under the coil is large enough for this to happen.

In this paper, we computationally analyze TMS-induced effects both on a neural population level and on a single neuron level. Firstly, we demonstrate that the dynamics of a simple excitatory-inhibitory balanced network well explains the temporal properties of visual percept suppression induced by a single pulse TMS. Secondly, we demonstrate that sustained inhibitory effect by a subthreshold TMS is reproduced by the network model, but not by an isolated single neuron model. Finally, we propose plausible neural mechanisms underlying TMS-induced interference with coordinated neural activities in the cortical network.

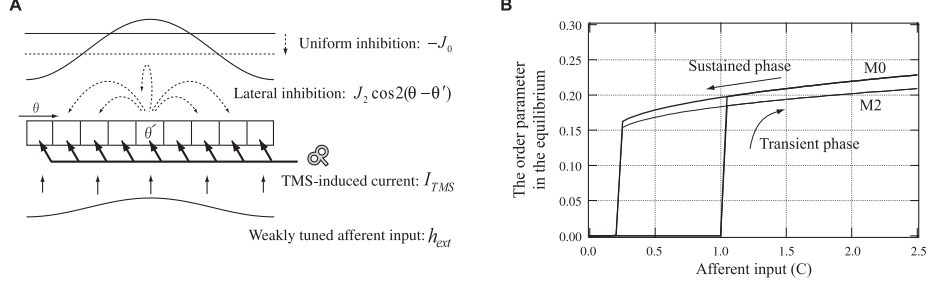

Figure 1: A) The network architecture. TMS was delivered to all neurons uniformly and simultaneously. B) The bistability in the network. The afferent input consisted of a suprathreshold transient and subthreshold sustained component leads the network into the bistable regime. The parameters used here are $\epsilon = 0.1, \beta = 0.25, J_0 = 73, J_2 = 110$, and $T = 1$.

## 2    Methods

### 2.1    TMS on neural population

#### 2.1.1    Network model for feature selectivity

We employed a simple excitatory-inhibitory balanced network model that is well analyzed as a model for a sensory feature detector system [3] (Fig. 1A):

$$\tau_m \frac{d}{dt} m(\theta, t) = -m(\theta, t) + g[h(\theta, t)] \tag{1}$$

$$h(\theta, t) = \int_{-\frac{\pi}{2}}^{\frac{\pi}{2}} \frac{d\theta'}{\pi} J(\theta - \theta') m(\theta', t) + h_{ext}(\theta, t) \tag{2}$$

$$J(\theta - \theta') = -J_0 + J_2 \cos 2(\theta - \theta') \tag{3}$$

$$h_{ext}(\theta, t) = c(t)[1 - \epsilon + \epsilon \cos 2(\theta - \theta_0)] \tag{4}$$

Here, $m(\theta, t)$ is the activity of neuron $\theta$ and $\tau_m$ is the microscopic characteristic time analogous to the membrane time constant of a neuron (Here we set $\tau_m = 10$ ms). $g[h]$ is a quasi-linear output function,

$$g[h] = \begin{cases} 0 & (h < T) \\ \beta(h - T) & (T \leq h < T + 1/\beta) \\ 1 & (h \geq T + 1/\beta) \end{cases} \tag{5}$$

where $T$ is the threshold of the neuron, $\beta$ is the gain factor, and $h(\theta, t)$ is the input to neuron $\theta$. For simplicity, we assume that $m(\theta, t)$ has a periodic boundary condition $(-\pi/2 \leq \theta \leq \pi/2)$, and the connections of each neuron are limited to this periodic range.

$\theta_0$ is a stimulus feature to be detected, and the afferent input, $h_{ext}(\theta, t)$, has its maximal amplitude $c(t)$ at $\theta = \theta_0$. We assume a static visual stimulus so that $\theta_0$ is constant during the stimulation (Hereafter we set $\theta_0 = 0$). $\epsilon$ is an afferent tuning coefficient, describing how the afferent input to the target population has already been localized around $\theta_0$ $(0 \leq \epsilon \leq 1/2)$.

The synaptic weight from neuron $\theta$ to $\theta'$, $J(\theta - \theta')$, consists of the uniform inhibition $J_0$ and a feature-specific interaction $J_2$. $J_0$ increases an effective threshold and regulates the whole network activity through all-to-all inhibition. $J_2$ facilitates neurons neighboring in the feature space and suppresses distant ones through a cosine-type connection weight.

Through these recurrent interactions, the activity profile of the network evolves and sharpens after the afferent stimulus onset.

The most intuitive and widely accepted example representable by this model is the orientation tuning function of the primary visual cortex [3][4][5]. Assuming that the coded feature is the orientation of a stimulus, we can regard $\theta$ as a neuron responding to angle $\theta$, $h_{ext}$ as an input from the lateral geniculate nucleus (LGN), and $J$ as a recurrent interaction in the primary visual cortex (V1).

Because the synaptic weight and afferent input have only the 0th and 2nd Fourier components, the network state can be fully described by the two order parameters $m_0$ and $m_2$, which are 0th- and 2nd-order Fourier coefficients of $m(\theta, t)$. The macroscopic dynamics of the network is thus derived by Fourier transformation of $m(\theta, t)$,

$$\tau_m \frac{d}{dt} m_0(t) = -m_0(t) + \int_{-\frac{\pi}{2}}^{\frac{\pi}{2}} \frac{d\theta}{\pi} g[h(\theta, t)] \tag{6}$$

$$\tau_m \frac{d}{dt} m_2(t) = -m_2(t) + \int_{-\frac{\pi}{2}}^{\frac{\pi}{2}} \frac{d\theta}{\pi} g[h(\theta, t)] \cos 2\theta \tag{7}$$

where $m_0(t)$ represents the mean activity of the entire network and $m_2(t)$ represents the degree of modulation of the activity profile of the network. $h(\theta, t)$ is also described by the order parameter,

$$h(\theta, t) = -J_0 m_0(t) + c(t)(1 - \epsilon) + (\epsilon c(t) + J_2 m_2(t)) \cos 2\theta \tag{8}$$

Substituting Eq.8 into Eq.6 and 7, the network dynamics can be calculated numerically.

### 2.1.2 TMS induction

We assumed that the TMS perturbation would be constant for all neurons in the network because the spatial extent of the neural population that we were dealing with is small compared with the spatial gradient of the induced electric field. Thus we modified the input function as $\hat{h}(\theta, t) = h(\theta, t) + I_{\text{TMS}}(t)$. Eq.6 to 8 were also modified accordingly by replacing $h$ with $\hat{h}$. Here we employ a simple rectangular input (amplitude: $I_{\text{TMS}}$, duration: $D_{\text{TMS}}$) as a TMS-like perturbation (see the middle graph of Fig. 2A).

### 2.1.3 Bistability and afferent input model

TMS applied to the occipital area after visual stimulus presentation typically suppresses its visual percept [6][7][8]. To determine whether the network model produces suppression similar to the experimental data, we applied a TMS-like perturbation at various timings after the afferent onset and examined whether the final state was suppressed or not. For this purpose, the network must hold two equilibria for the same afferent input condition and reach one of them depending on the specific timing and intensity of TMS. We thus chose proper sets of $\beta$, $J_0$, and $J_2$ that operated the network in the non-linear regime. In addition, we employed an afferent input model consisting of suprathreshold transient (amplitude: $A_t > T$, duration: $D_t$) and subthreshold sustained (amplitude: $A_s < T$) components (see the bottom graph of Fig. 2A). This is the simplest input model to lead the network into the bistable range (Fig. 1B), yet it still captures the common properties of neural signals in brain areas such as the LGN and visual cortex.

## 2.2 TMS on single neuron

### 2.2.1 Compartment model of cortical neuron

We also examined the effect of TMS on an isolated single neuron by using a compartment model of a neocortical neuron analyzed by Mainen and Sejnowski [9]. The model included

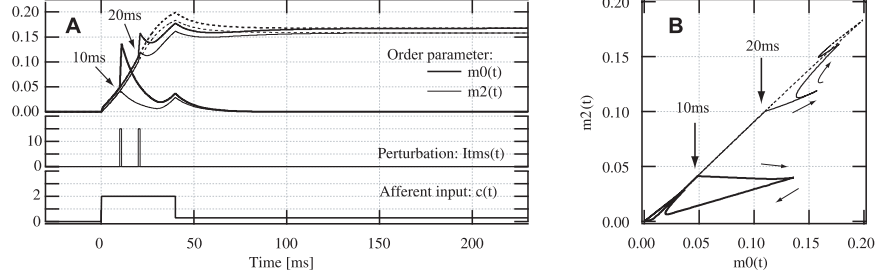

Figure 2: A) The time course of the order parameters, the perturbation, and the afferent input. B) The network state in the order parameter's plane. The network bifurcates depending on the induction timing of the perturbation and converges to either of the attractors. Two examples of TMS induction timing (10 and 20 ms after the afferent onset) are shown here. The dotted lines indicate the control condition without the perturbation in both graphs.

the following membrane ion channels: a low density of $Na^+$ channels in soma and dendrites and a high density in the axon hillock and the initial segment, fast $K^+$ channels in soma but not in dendrites, slow calcium- and voltage-dependent $K^+$ channels in soma and dendrites, and high-threshold $Ca^{2+}$ channels in soma and dendrites. We examined several types of cellular morphology as Mainen's report but excluded axonal compartments in order to evaluate the effect of induced current only from dendritic arborization. We injected a constant somatic current and observed a specific spiking pattern depending on morphology (Fig. 5).

### 2.2.2 TMS induction

There have been several reports on theoretically estimating the intracellular current induced by TMS [1][2][10]. Here we briefly describe a simple expression for the axial and transmembrane current induced by TMS. The electric field $E$ induced by a brief magnetic pulse is given by the temporal derivative of the magnetic vector potential $A$, i.e., $E(s,t) = -\partial A(s,t)/\partial t$. Suppose the spatial gradient of the induced magnetic field is so small compared to a single cellular dimension that $E$ can be approximated to be constant over all compartments. The simplest case is that one compartment has one distal and one proximal connection, in which the transmembrane current can be defined as the difference between the axial current going into and coming out of the adjacent compartment. The axial current between the adjacent compartment can be uniquely determined by distance and axial conductance between them (Fig. 5B),

$$I_a^{\text{TMS}}(j,k) = G_{jk} \int_{s_j}^{s_k} E(s) \cdot ds = G_{jk} E \cdot s_{jk}. \tag{9}$$

Hence the transmembrane current in the $k$-th compartment is,

$$I_m^{\text{TMS}}(k) = I_a^{\text{TMS}}(j,k) - I_a^{\text{TMS}}(k,l) = E \cdot (G_{jk} s_{jk} - G_{kl} s_{kl}). \tag{10}$$

Now we see that the important factors to produce a change in local membrane potential by TMS are the differences in axial conductance and position between adjacent compartments. As Nagarajan and Kamitani pointed out [1][2], if the cellular size is small, the heterogeneity of the local cellular properties (e.g. branching, ending, bending of dendrites, and change in dendrite diameter) could be crucial in inducing an intracellular current by TMS. A multiple branching formulation is easily obtained from Eq.10. For simplicity, the induced electric field was approximated as a rectangular pulse. The pulse's duration was set to be 1 ms, as in the network model, and the amplitude was varied within a physically valid range according to the numerical experiment's conditions.

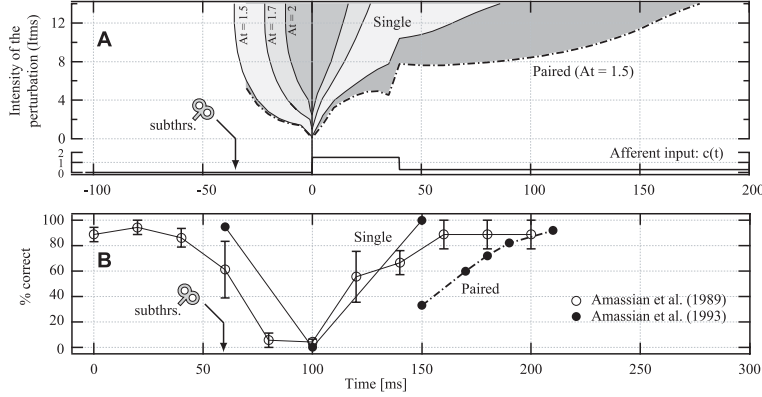

Figure 3: A) The minimum intensity of the suppressive perturbation in our model (solid line for single- and dashed line for paired-pulse). The width of each curve indicates the suppressive latency range for a particular intensity of the perturbation (e.g. if $A_t = 1.5$ and $I_{\mathrm{TMS}} = 12$, the network is suppressed during -35.5 to 64.2 ms for a single pulse case; thus the suppressive latency range is 99.7 ms.) B) Experimental data of suppressive effect on a character recognition task replotted and modified from [7] and [11]. Both graph A and B equivalently indicate the susceptibility to TMS at the particular timing. To compare the absolute timing, the model results must be biased with the proper amount of delay in neural signal transmission given to the target neural population because these are measured from the timing of afferent signal arrival, not from the onset of the visual stimulus presentation.

## 3 Results

### 3.1 Temporally selective suppression of neural population

The time course of the order parameters are illustrated in Fig. 2A. The network state can be also depicted as a point on a two-dimensional plane of the order parameters (Fig. 2B). Because TMS was modeled as a uniform perturbation, the mean activity, $m_0$, was transiently increased just after the onset of the perturbation and was followed by a decrease of both $m_0$ and $m_2$. This result was obtained regardless of the onset timing of the perturbation. The final state of the network, however, critically depended on the onset timing of the perturbation. It converged to either of the bistable states; the silent state in which the network activity is zero or the active state in which the network holds a local excitation. When the perturbation was applied temporally close to the afferent onset, the network was completely suppressed and converged to the silent state. On the other hand, when the perturbation was too early or too late from the afferent onset, the network was transiently perturbed but finally converged to the active state.

We could thus find the latency range during which the perturbation could suppress the network activity (Fig. 3A). The width of suppressive latency range increased with the amplitude of the perturbation and reached over 100 ms, which is comparable to typical experimental data of suppression of visual percepts by occipital TMS [6][7]. When we supplied a strong afferent input to the network, equivalent to a contrast increase in the visual stimulus, the suppressive latency range narrowed and shifted upward, and consequently, it became difficult to suppress the network activity without a strict timing control and larger amplitude of the perturbation. These results also agree with experiments using visual stimuli of various contrasts or visibilities [8][13]. The suppressive latency range consistently had a bell shape with the bottom at the afferent onset regardless of parameter changes, indicating that TMS works most suppressively at the timing when the afferent signal reaches the target

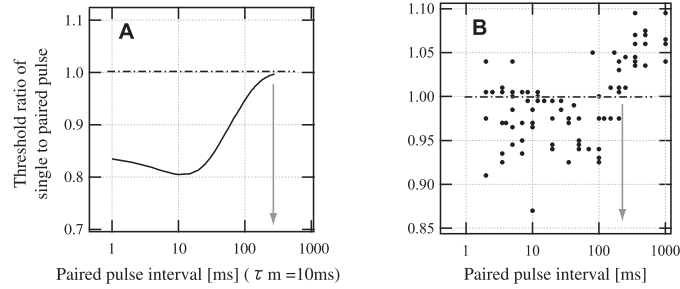

Figure 4: Threshold reduction by paired pulses in the steady state. A) Network model and B) experimental data of the phosphene threshold replotted from [12]. The dashed line indicates the threshold for a single pulse TMS.

neural population.

### 3.2 Sustained inhibition of neural population by subthreshold pulse

Multiple TMS pulses within a short interval, or repetitive TMS (rTMS), can evoke phosphene or visual deficits even though each single pulse fails to elicit any perceptible effect. This experimental fact suggests that a TMS pulse, even if it is a subthreshold one, induces a certain sustained inhibitory effect and reduces the next pulse's threshold to elicit perceptible interference.

We considered the effect of paired pulses on a neural population and determined the duration of the threshold reduction by a subthreshold TMS. Here we set the subthreshold level at the upper limit of intensity which could not suppress the network at the induction timing. For the steady state, the initial subthreshold perturbation significantly reduced the suppressive threshold for the subsequent perturbation; the original threshold level was restored to more than 100 ms after the initial TMS (Fig. 4A). The threshold slightly increased when the pulse interval was shorter than $\tau_m$. These results agree with experimental data of occipital TMS examining the relationship between phosphene threshold and the paired-pulse TMS interval [12] (Fig. 4B).

For the transient state, we also observed that the initial subthreshold perturbation, indicated by the arrow in Fig. 3A, significantly reduced the suppressive threshold for the subsequent perturbation, and consequently, the suppressive latency range was extended up to 60 ms (Fig. 3A). These results are consistent with Amassian's experimental results demonstrating that a preceding subthreshold TMS to the occipital cortex increased the suppressive latency range in a character recognition task [11] (Fig. 3B).

### 3.3 Transient inhibition of single neuron by subthreshold pulse

Next, we focus on the effect of TMS on a single neuron. Results from a layer V pyramidal cell are illustrated in Fig. 5. An intense perturbation could inhibit the spike train for over 100ms after a brief spike burst (Fig. 5C1). This sustained spike inhibition might be caused by mechanisms similar to after-hyperpolarization or adaptation because the intracellular concentration of $Ca^{2+}$ rapidly increased during the bursting period. These results are basically the same as Kamitani's report [1] using Poisson synapses as current inputs to the neuron. We tried several types of morphology and found that it was difficult to suppress their original spike patterns when the size of the neuron was small (e.g. stellate cell) or when the neuron initially showed spike bursts (e.g. pyramidal cell with more bushy dendritic arbors).

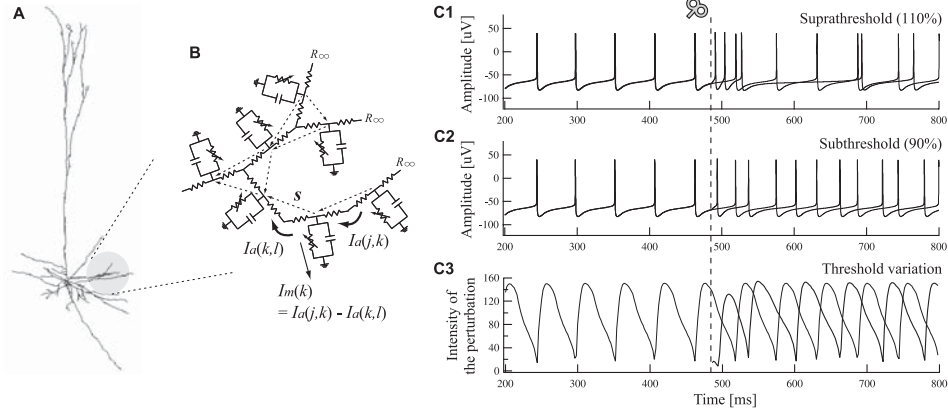

Figure 5: A) Layer V pyramidal cell. B) Compartment model of the neuron and the transmembrane current induced by TMS. C1, C2) The spike train perturbed by a suprathreshold and subthreshold TMS. C3) The temporal variation of the TMS threshold for inducing the spike inhibition. Thin lines in C1–C3 indicate the control condition without TMS.

Using a morphology whose spike train was most easily suppressed (i.e. a pyramidal cell in Fig. 5A), we determined whether a preceding subthreshold pulse could induce the sustained inhibitory effect. Here, the suppressive threshold was defined as the lowest intensity of the perturbation yielding a spike inhibitory period whose duration was more than 100 ms. The perturbation below the suppressive threshold caused the spike timing shift as illustrated in Fig. 5C2. In the single cell's case, the suppressive threshold highly depended on the relative timing within the spike interval and repeated its pattern periodically. In the initial spike interval from the subthreshold perturbation to the next spike, the suppressive threshold decreased but it recovered to the original level immediately after the next spike initiation (Fig. 5C3). This fast recovery of the suppressive threshold occurred regardless of the induction timing of the subthreshold perturbation, indicating that the sustained inhibitory effect by the preceding subthreshold perturbation lasted on the order of one (or two at most) spike interval, even with the most suppressible neuron model. The result is incomparably shorter than the experimental data as noted in Sec. 3.2, suggesting that it is impossible to attribute the neural substrates of the threshold reduction caused by the subthreshold pulse to only the membrane dynamics of a single neuron.

## 4  Discussion

This paper focused on the dichotomy to determine what is essential for TMS-induced suppression–a network or a single neuron? Our current answer is that the network is essential because the temporal properties of suppression observed in the neural population model were totally consistent with the experimental data. In a single neuron model, we can actually observe a spike inhibition whose duration is comparable to the silent period of the electromyogram induced by TMS on the motor cortex [14]; however, the degree of suppression is highly dependent on the property of the high-threshold $Ca^{2+}$ channel and is also very selective about the cellular morphology. In addition, the most critical point is that the sustained inhibitory effect of a subthreshold pulse cannot be explained by only the membrane mechanisms of a single neuron. These results indicate that TMS can induce a spike inhibition or a spike timing shift on a single neuron level, which yet seems not enough to explain the whole experimental data.

As Walsh pointed out [15], TMS is highly unlikely to evoke a coordinated activity pattern or to stimulate a specific functional structure with a fine spatial resolution in the target cortical area. Rather, TMS seems to induce a random activity irrespective of the existing neural activity pattern. This paper simply modeled TMS as a uniform perturbation simultaneously applied to all neurons in the network. Walsh's idea and our model are basically equivalent in that TMS gives a neural stimulation irrespective of the existing cortical activity evoked by the afferent input. Thus inactive parts of the network, or opponent neurons far from $\theta_0$, can be also activated by the perturbation if it is strong enough to raise such inactive neurons above the activation threshold, resulting in suppression of the original local excitation through lateral inhibitory connections. To suppress the network activity, TMS needs to be applied before the local excitation is built up and the inactive neurons are strongly suppressed. In the paired-pulse case, even though each TMS pulse was not strong enough to activate the suppressed neurons, the pre-activation by the preceding TMS can facilitate the subsequent TMS's effect if it is applied until the network restores its original activity pattern. These are the basic mechanisms of TMS-induced suppression in our model, by which the computational results are consistent with the various experimental data. In addition to our computational evidence, recent neuropharmacological studies demonstrated that GABAergic drugs [16] and hyperventilation environment [17] could modulate TMS effect, suggesting that transsynaptic inhibition via inhibitory interneuron might be involved in TMS-induced effects. All these facts indicate that TMS-induced neural interference is mediated by a transsynaptic network, not only by single neuron properties, and that inhibitory interactions in a neural population play a critical role in yielding neural interference and its temporal properties.

## Acknowledgments

We greatly appreciate our fruitful discussions with Dr. Yukiyasu Kamitani.

## References

[1] Y. Kamitani, V. Bhalodi, Y. Kubota, and S. Shimojo, Neurocomputing **38-40**, 697 (2001).

[2] S. Nagarajan, D. Durand, and E. Warman, IEEE Trans Biomed Eng **40**, 1175 (1993).

[3] R. Ben-Yishai, R. Bar-Or, and H. Sompolinsky, Proc Natl Acad Sci USA **92**, 3844 (1995).

[4] H. Sompolinsky and R. Shapley, Curr Opin Neurobiol **7**, 514 (1997).

[5] D. Somers, S. Nelson, and M. Sur, J Neurosci **15**, 5448 (1995).

[6] Y. Kamitani and S. Shimojo, Nat Neurosci **2**, 767 (1999).

[7] V. Amassian, R. Cracco, P. Maccabee, J. Cracco, A. Rudell, and L. Eberle, Electroencephalogr Clin Neurophysiol **74**, 458 (1989).

[8] T. Kammer and H. Nusseck, Neuropsychologia **36**, 1161 (1998).

[9] Z. Mainen and T. Sejnowski, Nature **382**, 363 (1996).

[10] B. Roth and P. Basser, IEEE Trans Biomed Eng **37**, 588 (1990).

[11] V. Amassian, P. Maccabee, R. Cracco, J. Cracco, A. Rudell, and E. L, Brain Res **605**, 317 (1993).

[12] P. Ray, K. Meador, C. Epstein, D. Loring, and L. Day, J Clin Neurophysiol **15**, 351 (1998).

[13] V. Amassian, R. Cracco, P. Maccabee, and J. Cracco, *Handbook of Transcranial Magnetic Stimulation* (Arnold Publisher, 2002), chap. 30, pp. 323–34.

[14] M. Inghilleri, A. Berardelli, G. Cruccu, and M. Manfredi, J Physiol **466**, 521 (1993).

[15] V. Walsh and A. Cowey, Nat Rev Neurosci **1**, 73 (2000).

[16] U. Ziemann, J. Rothwell, and M. Ridding, J Physiol **496.3**, 873 (1996).

[17] A. Priori, A. Berardelli, B. Mercuri, M. Inghilleri, and M. Manfredi, Electroencephalogr Clin Neurophysiol **97**, 69 (1995).
